# Statistical Debugging of Sampled Programs

**Alice X. Zheng**
EE Division
UC Berkeley
alicez@cs.berkeley.edu

**Michael I. Jordan**
CS Division and Department of Statistics
UC Berkeley
jordan@cs.berkeley.edu

**Ben Liblit**
CS Division
UC Berkeley
liblit@cs.berkeley.edu

**Alex Aiken**
CS Division
UC Berkeley
aiken@cs.berkeley.edu

## Abstract

We present a novel strategy for automatically debugging programs given sampled data from thousands of actual user runs. Our goal is to pinpoint those features that are most correlated with crashes. This is accomplished by maximizing an appropriately defined utility function. It has analogies with intuitive debugging heuristics, and, as we demonstrate, is able to deal with various types of bugs that occur in real programs.

## 1   Introduction

No software is perfect, and debugging is a resource-consuming process. Most users take software bugs for granted, and willingly run buggy programs every day with little complaint. In some sense, these user runs of the program are the ideal test suite any software engineer could hope for. In an effort to harness the information contained in these field tests, companies like Netscape/Mozilla and Microsoft have developed automated, opt-in feedback systems. User crash reports are used to direct debugging efforts toward those bugs which seem to affect the most people.

However, we can do much more with the information users may provide. Even if we collect just a little bit of information from every user run, successful or not, we may end up with enough information to automatically pinpoint the locations of bugs. In earlier work [1] we present a program sampling framework that collects data from users at minimal cost; the aggregated runs are then analyzed to isolate the bugs. Specifically, we learn a classifier on the data set, regularizing the parameters so that only the few features that are highly predictive of the outcome have large non-zero weights.

One limitation of this earlier approach is that it uses different methods to deal with different types of bugs. In this paper, we describe how to design a single classification utility function that integrates the various debugging heuristics. In particular, determinism of some features is a significant issue in this domain, and an additional penalty term for false positives is included to deal with this aspect. Furthermore, utility levels, while subjective, are robust: we offer simple guidelines for their selection, and demonstrate that results remain stable and strong across a wide range of reasonable parameter settings.

We start by briefly describing the program sampling framework in Section 2, and present the feature selection framework in Section 3. The test programs and our data set are described in Section 4, followed by experimental results in Section 5.

## 2  Program Sampling Framework

Our approach relies on being able to collect information about program behavior at runtime. To avoid paying large costs in time or space, we sparsely sample the program's runtime behavior. We scatter a large number of checks in the program code, but do not execute all of them during any single run. The sampled results are aggregated into counts which no longer contain chronology information but are much more space efficient.

To catch certain types of bugs, one asks certain types of questions. For instance, function call return values are good sanity checks which many programmers neglect. Memory corruption is another common class of bugs, for which we may check whether pointers are within their prescribed ranges. We add a large set of commonly useful assertions into the code, most of which are wild guesses which may or may not capture interesting behavior. At runtime, the program tosses a coin (with low heads probability) independently for each assertion it encounters, and decides whether or not to execute the assertion.

However, while it is not expensive to generate a random coin toss, doing so separately for each assertion would incur a very large overhead; the program will run even slower than just executing every assertion. The key is to combine coin tosses. Given i.i.d. Bernoulli random variables with success probability $h$, the number of trials it takes until the first success is a geometric random variable with probability $P(n) = (1 - h)^{n-1}h$. Instead of tossing a Bernoulli coin $n$ times, we can generate a geometric random variable to be used as a countdown to the next sample. Each assertion decrements this countdown by 1; when it reaches 0, we perform the assertion and generate another geometric random variable.[1]

However, checking to see if the counter has reached 0 at every assertion is still an expensive procedure. For further code optimization, we analyze each contiguous acyclic code region (loops- and recursion-free) at compile time and count the maximum number of assertions on any path through that region. Whenever possible, the generated code decrements in bulk, and takes a fast path that skips over the individual checks within a contiguous code region using just a single check against this maximum threshold.

Samples are taken in chronological order as the program runs. Useful as it might be, it would take a huge amount of space to record this information. To save space, we instead record only the counts of how often each assertion is found to be true or false. When the program finishes, these counts, along with the program exit status, are sent back to the central server for further analysis.

The program sampling framework is a non-trivial software analysis effort. Interested readers may refer to [1] for a more thorough treatment of all the subtleties, along with detailed analyses of performance impact at different sampling rates.

## 3  Classification and Feature Selection

In the hopes of catching a wide range of bugs, we add a large number of rather wild guesses into the code. Having cast a much bigger net than what we may need, the next step is to identify the relevant features. Let crashes be labeled with an output of 1, and successes labeled with 0. Knowing the final program exit status (crashed or successful) leaves us in

a classification setting. However, our primary goal is that of feature selection [2]. Good feature selection should be corroborated by classification performance, though in our case, we only care about features that correctly predict one of the two classes. Hence, instead of working in the usual maximum likelihood setting for classification and regularization, we define and maximize a more appropriate utility function. Ultimately, we will see that the two are not wholly unrelated.

It has been noted that the goals of variable and feature selection do not always coincide with that of classification [3]. Classification is but the means to an end. As we demonstrate in Section 5, good classification performance assures the user that the system is working correctly, but one still has to examine the selected features to see that they make sense.

### 3.1 Some characteristics of the problem

We concentrate on isolating the bugs that are caused by the occurrence of a small set of features, i.e. assertions that are always true when a crash occurs.[2] We want to identify the predicate counts that are positively correlated with the program crashing. In contrast, we do not care much about the features that are highly correlated with successes. This makes our feature selection an inherently one-sided process.

Due to sampling effects, it is quite possible that a feature responsible for the ultimate crash may not have been observed in a given run. This is especially true in the case of "quick and painless" deaths, where a program crashes very soon after the actual bug occurs. Normally this would be an easy bug to find, because one wouldn't have to look very far beyond the crashing point at the top of the stack. However, this is a challenge for our approach, because there may be only a single opportunity to sample the buggy feature before the program dies. Thus many crashes may have an input feature profile that is very similar to that of a successful run. From the classification perspective, this means that false negatives are quite likely.

At the other end of the spectrum, if we are dealing with a *deterministic bug*[3], false positives should have a probability of zero: if the buggy feature is observed to be true, then the program has to crash; if the program did not crash, then the bug must not have occurred. Therefore, for a deterministic bug, any false positives during the training process should incur a much larger penalty compared to any false negatives.

### 3.2 Designing the utility function

Let $(x, y)$ denote a data point, where $x$ is an input vector of non-negative integer counts, and $y \in \{0, 1\}$ is the output label. Let $f(x; \theta)$ denote a classifier with parameter vector $\theta$. There are four possible prediction outcomes: $y = 1$ and $f(x; \theta) = 1$, $y = 0$ and $f(x; \theta) = 0$, $y = 1$ and $f(x; \theta) = 0$, and $y = 0$ and $f(x; \theta) = 1$. The last two cases represent false negative and false positive, respectively. In the general form of utility maximization for classification (see, e.g., [4]), we can define separate utility functions for each of the four cases, and maximize the sum of the expected utilities:

$$\max_{\theta} \mathbb{E}_{P(Y|x)} U(Y, x; \theta), \tag{1}$$

$$\text{where} \quad U(Y, x; \theta) = u_1(x; \theta) Y \mathbb{I}_{\{f(x;\theta)=1\}} + u_2(x; \theta) Y \mathbb{I}_{\{f(x;\theta)=0\}}$$
$$+ u_3(x; \theta)(1 - Y) \mathbb{I}_{\{f(x;\theta)=0\}} + u_4(x; \theta)(1 - Y) \mathbb{I}_{\{f(x;\theta)=1\}} + v(\theta), \tag{2}$$

and where $\mathbb{I}_W$ is the indicator function for event $W$. The $u_i(x;\theta)$ functions specify the utility of each case. $v(\theta)$ is a regularization term, and can be interpreted as a prior over the classifier parameters $\theta$ in the Bayesian terminology.

We can approximate the distribution $P(Y|x)$ simply by its empirical distribution, $P(Y = 1|x;\theta) := \hat{P}(Y = 1|x) = y$. The actual distribution of input features $X$ is determined by the software under examination, hence it is difficult to specify and highly non-Gaussian. Thus we need a discriminative classifier. Let $z = \theta^T x$, where the $x$ vector is now augmented by a trailing 1 to represent the intercept term.[4] We use the logistic function $\mu(z)$ to model the class conditional probability:

$$P(Y = 1|x) \quad := \quad \mu(z) = 1/(1 + e^{-z}). \tag{3}$$

The decision boundary is set to $1/2$, so that $f(x;\theta) = 1$ if $\mu(z) > 1/2$, and $f(x;\theta) = 0$ if $\mu(z) \leq 1/2$. The regularization term is chosen to be the $\ell_1$ norm of $\theta$, which has the effect of driving most $\theta_i$'s to zero: $v(\theta) := -\lambda|\theta|_1^1 = -\lambda\sum_i|\theta_i|$. To slightly simplify the formula, we choose the same functional form for $u_1$ and $u_2$, but add an extra penalty term for false positives:

$$u_1(x;\theta) \quad := \quad u_2(x;\theta) \quad := \quad \delta_1(\log_2\mu(x;\theta) + 1) \tag{4}$$

$$u_3(x;\theta) \quad := \quad \delta_2(\log_2(1 - \mu(x;\theta)) + 1) \tag{5}$$

$$u_4(x;\theta) \quad := \quad \delta_2(\log_2(1 - \mu(x;\theta)) + 1) - \delta_3\theta^T x . \tag{6}$$

Note that the additive constants do not affect the outcome of the optimization; they merely ensure that utility at the decision boundary is zero. Also, we can fold any multiplicative constants of the utility functions into $\delta_i$, so the base of the $\log$ function is freely exchangeable. We find that the expected utility function is equivalent to:

$$\mathbb{E}\,U = \delta_1 y \log\mu + \delta_2(1 - y)\log(1 - \mu) - \delta_3\theta^T x(1 - y)\mathbb{I}_{\{\mu > 1/2\}} - \lambda\|\theta\|_1^1 . \tag{7}$$

When $\delta_1 = \delta_2 = 1$ and $\delta_3 = 0$, Eqn. (7) is akin to the Lasso [5] (standard logistic regression with ML parameter estimation and $\ell_1$-norm regularization). In general, this expected utility function weighs each class separately using $\delta_i$, and has an additional penalty term for false positives.

Parameter learning is done using stochastic (sub)gradient ascent on the objective function. Besides having desirable properties like fast convergence rate and space efficiency, such on-line methods also improve user privacy. Once the sufficient statistics are collected, the trial run can be discarded, thus obviating the need to permanently store any user's private data on a central server.

Eqn. (7) is concave in $\theta$, but the $\ell_1$ norm and the indicator function are non-differentiable at $\theta_i = 0$ and $\theta^T x = 0$, respectively. This can be handled by subgradient ascent methods[5]. In practice, we jitter the solution away from the point of non-differentiability by taking a very small step along any subgradient. This means that none of the $\theta_i$'s will ever be exactly zero. But this does not matter since weights close enough to zero are essentially taken as zero. Only the few features with the most positive weights are selected at the end.

### 3.3 Interpretation of the utility functions

Let us closely examine the utility functions defined in Eqns. (4)–(6). For the case of $Y = 1$, Fig. 1(a) plots the function $\log_2\mu(z) + 1$. It is positive when $z$ is positive, and approaches

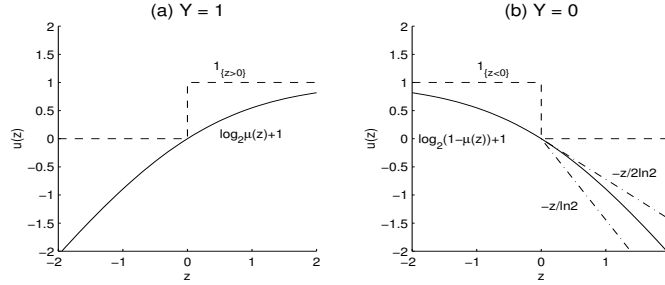

Figure 1: (a) Plot of the true positive indicator function and the utility function $\log_2 \mu(z) + 1$. (b) Plot of the true negative indicator function, utility function $\log_2(1 - \mu(z)) + 1$, and its asymptotic slopes $-z/\log 2$ and $-z/2\log 2$.

1 as $z$ approaches $+\infty$. It is a crude but smooth approximation of the indicator function for a true positive, $y\mathbb{I}_{\{\mu > 1/2\}}$. On the other hand, when $z$ is negative, the utility function is negative, acting as a penalty for false negatives. Similarly, Fig. 1(b) plots the utility functions for $Y = 0$. In both cases, the utility function has an upper bound of 1, so that the effect of correct classifications is limited. On the other hand, incorrect classifications are undesirable, thus their penalty is an unbounded (but slowly deceasing) negative number.

Taking the derivative $\frac{d}{dz}\log_2(1 - \mu(z) + 1) = -\mu(z)/\log 2$, we see that, when $z$ is positive, $-1 \leq -\mu(z) \leq -1/2$, so $\log_2(1 - \mu(z)) + 1$ is sandwiched between two linear functions $-z/\log 2$ and $-z/2\log 2$. It starts off being closer to $-z/2\log 2$, but approaches $-z/\log 2$ asymptotically (see Fig. 1(b)). Hence, when the false positive is close to the decision boundary, the additional penalty of $\theta^T x = z$ in Eqn. (6) is larger than the default false positive penalty, though the two are asymptotically equivalent.

Let us turn to the roles of the multiplicative weights. $\delta_1$ and $\delta_2$ weigh the relative importance of the two classes, and can be used to deal with imbalanced training sets where one class is disproportionately larger than the other [7]. Most of the time a program exits successfully without crashing, so we have to deal with having many more successful runs than crashed runs (see Section 5). Furthermore, since we really only care about predicting class 1, increasing $\delta_1$ beyond an equal balance of the two data sets could be beneficial for feature selection performance. Finally, $\delta_3$ is the knob of determinism: if the bug is deterministic, then setting $\delta_3$ to a large value will severely penalize false positives; if the bug is not deterministic, then a small value for $\delta_3$ affords the necessary slack to accommodate runs which should have failed but did not. As we shall see in Section 5, if the bug is truly deterministic, then the quality of the final features selected will be higher for large $\delta_3$ values.

In a previous paper [1], we outlined some simple feature elimination heuristics that can be used in the case of a deterministic bug. ⟨Elimination by universal falsehood⟩ discards any counter that is always zero, because it likely represents an assertion that can never be true. This is a very common data preprocessing step. ⟨Elimination by lack of failing example⟩ discards any counter that is zero on all crashes, because what never happens cannot have caused the crash. ⟨Elimination by successful counterexample⟩ discards any counter that is non-zero on any successful run, because these are assertions that can be true without a subsequent program failure. In our model, if a feature $x_i$ is never positive for any crashes, then its associated weight $\theta_i$ will only decrease in the maximization process. Thus it will not be selected as a crash-predictive feature. This handles ⟨elimination by lack of failing example⟩. Also, if a heavily weighted feature $x_i$ is positive on a successful run in the training set, then the classifier is more likely to result in a false positive. The false positive penalty term will then decrease the weight $\theta_i$, so that such a feature is unlikely to be chosen at the end. Thus utility maximization also handles ⟨elimination by successful counterexample⟩. The model we derive here, then, neatly subsumes the ad hoc elimination heuristics used in our earlier work.

## 4   Two Case Studies

As examples, we present two cases studies of C programs with bugs that are at the opposite ends of the determinism spectrum. Our deterministic example is `ccrypt`, a small encryption utility. `ccrypt-1.2` is known to contain a bug that involves overwriting existing files. If the user responds to a confirmation prompt with `EOF` rather than `yes` or `no`, `ccrypt` consistently crashes. Our non-deterministic example is GNU `bc-1.06`, the Unix command line calculator tool. We find that feeding `bc` nine megabytes of random input causes it to crash roughly one time in four while calling `malloc()` — a strong indication of heap corruption. Such bugs are inherently difficult to fix because they are inherently non-deterministic: there is no guarantee that a mangled heap will cause a crash soon or indeed at all.

`ccrypt`'s sensitivity to `EOF` inputs suggests that the problem has something to do with its interactions with standard file operations. Thus, randomly sampling function return values may identify key operations close to the bug. Our instrumented program adds instrumentation after each function call to sample and record the number of times the return value is negative, zero, or positive. There are 570 call sites of interest, for $570 \times 3 = 1710$ counters. In lieu of a large user community, we generate many runs artificially using reasonable inputs. Each run uses a randomly selected set of present or absent files, randomized command line flags, and randomized responses to `ccrypt` prompts including the occasional `EOF`. We have collected 7204 trial runs at a sampling rate of $1/100$, 1162 of which result in a crash. 6516 ($\approx 90\%$) of these trial runs are randomly selected for training, and the remaining 688 held aside for cross-validation. Out of the 1710 counter features, 1542 are constant across all runs, leaving 168 counters to be considered in the training process.

In the case of `bc`, we are interested in the behavior of all pointers and buffers. All pointers and array indices are scalars, hence we compare all pairs of scalar values. At any direct assignment to a scalar variable $a$, we identify all other variables $b_1, b_2, \ldots, b_n$ of the same type that are also in scope. We record the number of times that $a$ is found to be greater than, equal to, or less than each $b_i$. Additionally, we compare each pointer to the `NULL` value. There are 30150 counters in all, of which 2908 are not constant across all runs. Our `bc` data set consists of 3051 runs with distinct random inputs at a sampling rate of $1/1000$. 2729 of these runs are randomly chosen as training set, 322 for the hold-out set.

## 5   Experimental Results

We maximize the utility function in Eqn. (7) using stochastic subgradient ascent with a learning rate of $10^{-5}$. In order to make the magnitude of the weights $\theta_i$ comparable to each other, the feature values are shifted and scaled to lie between $[0, 1]$, then normalized to have unit variance. There are four learning parameters, $\delta_1$, $\delta_2$, $\delta_3$, and $\lambda$. Since only their relative scale is important, the regularization parameter $\lambda$ can be set to some fixed value (we use $0.1$). For each setting of $\delta_i$, the model is set to run for 60 iterations through the training set, though the process usually converges much sooner. For `bc`, this takes roughly 110 seconds in MATLAB on a 1.8 GHz Pentium 4 CPU with 1 GB of RAM. The smaller `ccrypt` dataset requires just under 8 seconds.

The values of $\delta_1$, $\delta_2$, and $\delta_3$ can all be set through cross-validation. However, this may take a long time, plus we would like to leave the ultimate control of the values to the users of this tool. The more important knobs are $\delta_1$ and $\delta_3$: the former controls the relative importance of classification performance on crashed runs, the latter adjusts the believed level of determinism of the bug. Here are some guidelines for setting $\delta_1$ and $\delta_3$ that we find to work well in practice. (1) In order to counter the effects of imbalanced datasets, the ratio of $\delta_1/\delta_2$ should be at least around the range of the ratio of successful to crashed runs. This is especially crucial for the `ccrypt` data set, which contains roughly 32 successful runs for every crash. (2) $\delta_3$ should not be higher than $\delta_1$, because it is ultimately more important

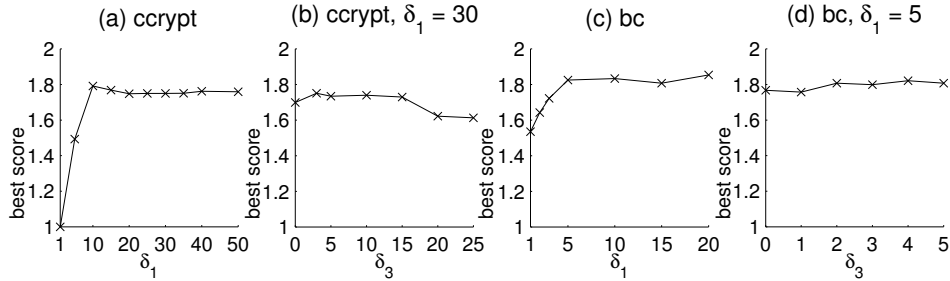

Figure 2: (a,b) Cross-validation scores for the `ccrypt` data set; (c,d) Cross-validation scores for the `bc` data set. All scores shown are the maximum over free parameters.

to correctly classify crashes than to not have any false positives.

As a performance metric, we look at the hold-out set confusion matrix and define the score as the sum of the percentages of correctly classified data points for each class. Fig. 2(a) shows a plot of cross-validation score (maximum over a number of settings for $\delta_2$ and $\delta_3$) for the `ccrypt` data set at various $\delta_1$ values. It is apparent from the plot that any $\delta_1$ values in the range of $[10, 50]$ are roughly equivalent in terms of classification performance. Specifically, for the case of $\delta_1 = 30$ (which is around the range suggested by Guideline 1 above), Fig. 2(b) shows the cross-validation scores plotted against different values for $\delta_3$. In this case, as long as $\delta_3$ is in the rough range of $[3, 15]$, the classification performance remains the same.[6]

Furthermore, settings for $\delta_1$ and $\delta_3$ that are safe for classification also select high quality features for debugging. The "smoking gun" which directly indicates the `ccrypt` bug is:

```
traverse.c:122:   xreadline() return value == 0
```

This call to `xreadline()` returns 0 if the input terminal is at `EOF`. In all of the above mentioned safe settings for $\delta_1$ and $\delta_3$, this feature is returned as the top feature. The rest of the higher ranked features are sufficient, but not necessary, conditions for a crash. The only difference is that, in more optimal settings, the separation between the top feature and the rest can be as large as an order of magnitude; in non-optimal settings (classification score-wise), the separation is smaller.

For `bc`, the classification results are even less sensitive to the particular settings of $\delta_1$, $\delta_2$, and $\delta_3$. (See Fig. 2(c,d).) The classification score is roughly constant for $\delta_1 \in [5, 20]$, and for a particular value of $\delta_1$, such as $\delta_1 = 5$, the value of $\delta_3$ has little impact on classification performance. This is to be expected: the bug in `bc` is non-deterministic, and therefore false positives do indeed exist in the training set. Hence any small value for $\delta_3$ will do.

As for the feature selection results for `bc`, for all reasonable parameter settings (and even those that do not have the best classification performance), the top features are a group of correlated counters that all point to the index of an array being abnormally big. Below are the top five features for $\delta_1 = 10, \delta_2 = 2, \delta_3 = 1$:

```
1. storage.c:176:   more_arrays():   indx > optopt
2. storage.c:176:   more_arrays():   indx > opterr
3. storage.c:176:   more_arrays():   indx > use_math
4. storage.c:176:   more_arrays():   indx > quiet
5. storage.c:176:   more_arrays():   indx > f_count
```

These features immediately point to line 176 of the file `storage.c`. They also indicate that the variable `indx` seems to be abnormally big. Indeed, `indx` is the array index that runs over the actual array length, which is contained in the integer variable `a_count`. The program may crash long after the first array bound violation, which means that there are many opportunities for the sampling framework to observe the abnormally big value of `indx`. Since there are many comparisons between `indx` and other integer variables, there is a large set of inter-correlated counters, any subset of which may be picked by our algorithm as the top features. In the training run shown above, the smoking gun of `indx > a_count` is ranked number 8. But in general its rank could be much smaller, because the top features already suffice for predicting crashes and pointing us to the right line in the code.

## 6   Conclusions and Future Work

Our goal is a system that automatically pinpoints the location of bugs in widely deployed software. We tackle different types of bugs using a custom-designed utility function with a "determinism level" knob. Our methods are shown to work on two real-world programs, and are able to locate the bugs in a range of parameter settings.

In the real world, programs contain not just one, but many bugs, which will not be distinctly labeled in the set of crashed runs. It is difficult to tease out the different failure modes through clustering: it relies on macro-level usage patterns, as opposed to the microscopic difference between failures. In on-going research, we are extending our approach to deal with the problem of multiple bugs in larger programs. We are also working on modifying the program sampling framework to allow denser sampling in more important regions of the code. This should alleviate the sparsity of features while reducing the number of runs required to yield useful results.

### Acknowledgments

This work was supported in part by ONR MURI Grant N00014-00-1-0637; NASA Grant No. NAG2-1210; NSF Grant Nos. EIA-9802069, CCR-0085949, ACI-9619020, and IIS-9988642; DOE Prime Contract No. W-7405-ENG-48 through Memorandum Agreement No. B504962 with LLNL.

## Footnotes

[1]The sampling density $h$ controls the tradeoff between runtime overhead and data sparsity. It is set to be small enough to have tolerable overhead, which then requires more runs in order to alleviate the effects of sparsity. This is not a problem for large programs like Mozilla and Windows with thousands of crash reports a day.

[2]There are bugs that are caused by non-occurrence of certain events, such as forgotten initializations. We do not focus on this type of bugs in this paper.

[3]A bug is *deterministic* if it crashes the program every time it is observed. For example, dereferencing a null pointer would crash the program without exception. Note that this notion of determinism is data-dependent: it is always predicated on the trial runs that we have seen.

[4]Assuming that the more abnormalities there are, the more likely it is for the program to crash, it is reasonable to use a classifier based on a linear combination of features.

[5]Subgradients are a generalization of gradients that are also defined at non-differentiable points. A subgradient for a convex function is any sublinear function pivoted at that point, and minorizing the entire convex function. For convex (concave) optimization, any subgradient is a feasible descent (ascent) direction. For more details, see, e.g., [6].

[6]In Fig. 2(b), the classification performance for $\delta_1 = 30$ and $\delta_3 = 0$ is deceptively high. In this case, the best $\delta_2$ value is 5, which offsets the cross-validation score by increasing the number of predicted non-crashes, at the expense of worse crash-prediction performance. The top feature becomes a necessary but not sufficient condition for a crash – a false positive-inducing feature! Hence the lesson is that if the bug is believed to be deterministic then $\delta_3$ should always be positive.

## References

[1] B. Liblit, A. Aiken, A. X. Zheng, and M. I. Jordan. Bug isolation via remote program sampling. In *ACM SIGPLAN PLDI 2003*, 2003.

[2] A. Blum and P. Langley. Selection of relevant features and examples in machine learning. *Artificial Intelligence*, 97(1-2):245–271, 1997.

[3] I. Guyon and A. Elisseeff. An introduction to variable and feature selection. *Journal of Machine Learning Research*, 3:1157–1182, March 2003.

[4] E. L. Lehmann. *Testing Statistical Hypotheses*. John Wiley & Sons, 2nd edition, 1986.

[5] T. Hastie, R. Tibshirani, and J. Friedman. *The Elements of Statistical Learning*. Springer–Verlag, 2001.

[6] J.-B. Hiriart-Urruty and C. Lemarechal. *Convex Analysis and Minimization Algorithms*, volume II. Springer–Verlag, 1993.

[7] N. Japkowicz and S. Stephen. The class imbalance problem: a systematic study. *Intelligent Data Analysis Journal*, 6(5), November 2002.
